# THE HOPFIELD MODEL WITH MULTI-LEVEL NEURONS

Michael Fleisher
Department of Electrical Engineering
Technion - Israel Institute of Technology
Haifa 32000, Israel

**ABSTRACT**

The Hopfield neural network model for associative memory is generalized. The generalization replaces two state neurons by neurons taking a richer set of values. Two classes of neuron input output relations are developed guaranteeing convergence to stable states. The first is a class of "continuous" relations and the second is a class of allowed quantization rules for the neurons. The information capacity for networks from the second class is found to be of order $N^3$ bits for a network with $N$ neurons.

A generalization of the sum of outer products learning rule is developed and investigated as well.

## I. INTRODUCTION

The ability to perform collective computation in a distributed system of flexible structure without global synchronization is an important engineering objective. Hopfield's neural network [1] is such a model of associative content addressable memory.

An important property of the Hopfield neural network is its guaranteed convergence to stable states (interpreted as the stored memories). In this work we introduce a generalization of the Hopfield model by allowing the outputs of the neurons to take a richer set of values than Hopfield's original binary neurons. Sufficient conditions for preserving the convergence property are developed for the neuron input output relations. Two classes of relations are obtained. The first introduces neurons which simulate multi threshold functions, networks with such neurons will be called quantized neural networks (Q.N.N.). The second class introduces continuous neuron input output relations and networks with such neurons will be called continuous neural networks (C.N.N.).

In Section II, we introduce Hopfield's neural network and show its convergence property. C.N.N. are introduced in Section III and a sufficient condition for the neuron input output continuous relations is developed for preserving convergence. In Section IV, Q.N.N. are introduced and their input output relations are analyzed in the same manner as in III. In Section IV we look further at Q.N.N. by using the definition of information capacity for neural networks of [2] to obtain a tight asymptotic estimate of the capacity for a Q.N.N. with $N$ neurons. Section VI is a generalized sum of outer products learning for the Q.N.N. and section VII is the discussion.

## II. THE HOPFIELD NEURAL NETWORK

A neural network consists of $N$ pairwise connected neurons. The $i$'th neuron can be in one of two states: $X_i = -1$ or $X_i = +1$. The connections are fixed real numbers denoted by $W_{ij}$ (the connection from neuron $i$ to neuron $j$ ). Define the state vector $\underline{X}$ to be a binary vector whose $i$'th component corresponds to the state of the $i$'th neuron. Randomly and asynchronously, each neuron examines its input and decides its next output in the following manner. Let $t_i$ be the threshold voltage of the $i$'th neuron. If the weighted sum of the present other $N-1$ neuron outputs (which compose the $i$'th neuron input) is

greater or equal to $t_i$, the next $X_i (X_i^+)$ is $+1$, if not, $X_i^+$ is $-1$. This action is given in (1).

$$X_i^+ = sgn \ [ \ \sum_{j=1}^{N} W_{ij} X_j - t_i \ ] \tag{1}$$

We give the following theorem

Theorem 1 (of [1])

The network described with symmetric $(W_{ij} = W_{ji})$ zero diagonal $(W_{ii} = 0)$ connection matrix $W$ has the convergence property.

Proof

Define the quantity

$$E(\underline{X}) = -\frac{1}{2} \sum_{i}^{N} \sum_{j=1}^{N} W_{ij} X_i X_j + \sum_{i=1}^{N} t_i X_i \tag{2}$$

We show that $E(\underline{X})$ can only decrease as a result of the action of the network. Suppose that $X_k$ changed to $X_k^+ = X_k + \Delta X_k$, the resulting change in $E$ is given by

$$\Delta E = -\Delta X_k \ ( \ \sum_{j=1}^{N} W_{kj} X_j - t_k) \tag{3}$$

(Eq. (3) is correct because of the restrictions on $W$). The term in brackets is exactly the argument of the sgn function in (1) and therefore the signs of $\Delta X_k$ and the term in brackets is the same (or $\Delta X_k = 0$) and we get $\Delta E \leq 0$. Combining this with the fact that $E(\underline{X})$ is bounded shows that eventually the network will remain in a local minimum of $E(\underline{X})$. This completes the proof.

The technique used in the proof of Theorem 1 is an important tool in analyzing neural networks. A network with a particular underlying $E(\underline{X})$ function can be used to solve optimization problems with $E(\underline{X})$ as the object of optimization. Thus we see another use of neural networks.

## III. THE C.N.N.

We ask ourselves the following question: How can we change the sgn function in (1) without affecting the convergence property? The new action rule for the $i$ 'th neuron is

$$X_i^+ = f_i[\ \sum_{j=1}^{N} W_{ij}X_j\ ] \tag{4}$$

Our attention is focused on possible choices for $f_i(\cdot)$. The following theorem gives a part of the answer.

### Theorem 2

The network described by (4) (with symmetric zero diagonal $\mathbf{W}$) has the convergence property if $f_i(\cdot)$ are strictly increasing and bounded.

### Proof

Define

$$E(\underline{X}) = -\frac{1}{2} \sum_{i}^{N} \sum_{j}^{N} W_{ij}X_iX_j + \sum_{i=1}^{N} \int_{0}^{X_i} f_i^{-1}(u)du \tag{5}$$

We show as before that $E(\underline{X})$ can only decrease and since $E$ is bounded (because of the boundedness of $f_i$'s) the theorem is proved.

Using $g_i(X_i) = \int_{0}^{X_i} f_i^{-1}(u)du$ we have

$$\Delta E = -\Delta X_k\ [\ \sum_{i=1}^{N} W_{kj}X_j - \frac{g_k(X_k+\Delta X_k) - g(X_k)}{\Delta X_k}\ ] \tag{6}$$

Using the intermediate value theorem we get

$$\Delta E = -\Delta X_k\ [\ \sum_{j=1}^{N} W_{kj}X_j - g_k'(C)] = -\Delta X_k\ [f_k^{-1}(X_k+\Delta X_k) - f_k^{-1}(C)] \tag{7}$$

where $C$ is a point between $X_k$ and $X_k + \Delta X_k$. Now, if $\Delta X_k > 0$ we have $C \leq X_k + \Delta X_k => f_k^{-1}(C) \leq f_k^{-1}(X_k + \Delta X_k)$ and the term in brackets is greater or equal to zero $=> \Delta E \leq 0$. A similar argument holds for $\Delta X_k < 0$ (of course $\Delta X_k = 0 => \Delta E = 0$). This completes the proof.

Some remarks:

(a) Strictly increasing bounded neuron relations are not the whole class of relations conserving the convergence property. This is seen immediately from the fact that Hopfield's original model (1) is not in this class.

(b) The $E(\underline{X})$ in the C.N.N. coincides with Hopfield's continuous neural network [3]. The difference between the two networks lies in the updating scheme. In our C.N.N. the neurons update their outputs at the moments they examine their inputs while in [3] the updating is in the form of a set of differential equations featuring the time evolution of the network outputs.

(c) The boundedness requirement of the neuron relations results from the boundedness of $E(\underline{X})$. It is possible to impose further restrictions on $W$ resulting in unbounded neuron relations but keeping $E(\underline{X})$ bounded (from below). This was done in [4] where the neurons exhibit linear relations.

## IV. THE Q.N.N.

We develop the class of quantization rules for the neurons, keeping the convergence property. Denote the set of possible neuron outputs by $Y_o < Y_1 < ... < Y_n$ and the set of threshold values by $t_1 < t_2 < \cdots < t_n$ the action of the neurons is given by

$$X_i^+ = Y_l \quad \text{if} \quad t_l < \sum_{j=1}^{N} W_{ij} X_j \leq t_{l+1} \quad l = 0,...,n \tag{8}$$

and $t_o = -\infty, t_{n+1} = +\infty$.

The following theorem gives a class of quantization rules with the convergence property.

Theorem 3

Any quantization rule for the neurons which is an increasing step function that is

$$Y_0 < Y_1 < \cdots Y_n \; ; t_1 < \cdots < t_n \tag{9}$$

Yields a network with the convergence property (with a $\mathbf{W}$ symmetric and zero diagonal).

Proof

We proceed to prove.

Define

$$E(\underline{X}) = -\frac{1}{2} \sum_i^N \sum_{j=1}^N W_{ij} X_i X_j + \sum_{i=1}^N tG(X_i) + \sum_{i=1}^N dX_i \tag{10}$$

where $G(X)$ is a piecewise linear convex $U$ function defined by the relation

$$t \frac{G(Y_l) - G(Y_{l-1})}{Y_l - Y_{l-1}} + d = t_l \quad l = 1, \ldots, n \tag{11}$$

As before we show $\Delta E \leq 0$. Suppose a change occurred in $X_k$ such that $X_k = Y_{i-1}, X_k^+ = Y_i$. We then have

$$\Delta E = -\Delta X_k \left[ \sum_{j=1}^N W_{kj} X_j - t \frac{G(X_k^+) - G(X_k)}{\Delta X_k} - d \right] = -\Delta X_k \left[ \sum_{j=1}^N W_{kj} X_j - t_k \right] \leq 0 \tag{12}$$

A similar argument follows when $X_k = Y_i, X_k^+ = Y_{i-1} < X_k$. Any bigger change in $X_k$ (from $Y_i$ to $Y_j$ with $|i-j| > 1$) yields the same result since it can be viewed as a sequence of $|i-j|$ changes from $Y_i$ to $Y_j$ each resulting in $\Delta E \leq 0$. The proof is completed by noting that $\Delta X_k = 0 => \Delta E = 0$ and $E(\underline{X})$ is bounded.

Corollary

Hopfield's original model is a special case of (9).

## V. INFORMATION CAPACITY OF THE Q.N.N.

We use the definition of [2] for the information capacity of the Q.N.N.

Definition 1

The information capacity of the Q.N.N. (bits) is the $\log$ (Base 2) of the number of distinguishable networks of $N$ neurons. Two networks are distinguishable if observing the state transitions of the neurons yields different observations. For Hopfield's original model it was shown in [2] that the capacity $C$ of a network of $N$ neurons is bounded by $C \leq \log (2^{(N-1)^2})^N = 0(N^3)b$. It was also shown that $C \geq \Omega(N^3)b$ and thus is exactly of the order $N^3 b$. It is obvious that in our case (which contains the original model) we must have $C \geq \Omega(N^3)b$ as well (since the lower bound cannot decrease in this richer case). It is shown in the Appendix that the number of multi threshold functions of $N-1$ variables with $n+1$ output levels is at most $(n+1)^{N^2+N+1}$ since we have $N$ neurons there will be $((n+1)^{N^2+N+1})^N$ distinguishable networks and thus

$$C \leq \log ((n+1)^{N^2+N+1})^N = 0(N^3)b \tag{14}$$

oı as before, $C$ is exactly of $0(N^3)b$. In fact, the rise in $C$ is probably a factor of $0(\log_2 n)$ as can be seen from the upper bound.

## VI. "OUTER PRODUCT" LEARNING RULE

For Hopfield's original network with two state neurons (taking the values $\pm 1$) a natural and extensively investigated [ ],[ ],[ ] learning rule is the so called sum of outer products construction.

$$W_{ij} = \frac{1}{N} \sum_{l=1}^{K} X_i^l X_j^l \tag{15}$$

where $\underline{X}^1, \ldots, \underline{X}^K$ are the desired stable states of the network. A well-known result for (15) is that the asymptotic capacity $K$ of the network is

$$K = \frac{N-1}{4\log N} + 1 \qquad (16)$$

In this section we introduce a natural generalization of (15) and prove a similar result for the asymptotic capacity. We first limit the possible quantization rules to:

$$X_i = F(u_i) = \begin{cases} Y_o & t_1 > u_i \geq t_o \\ \cdot \\ \cdot \\ \cdot \\ Y_n & t_{n+1} > u_i \geq t_n \end{cases} \qquad (17)$$

with $Y_o < \cdots < Y_n$

$$t_j = \frac{1}{2}\left[Y_j + Y_{j-1}\right] \qquad j=1, \cdots n$$

$$t_o = -\infty \quad ; \quad t_{n+1} = \infty$$

with

(a)   $n+1$   is even
(b)   $\forall \, i \quad Y_i \neq 0$
(c)   $Y_i = -Y_{n-i} \quad i=0, \ldots, n$

Next we state that the desired stable vectors $\underline{X}^1, \cdots \underline{X}^K$ are such that each component is picked independently at random from $\{Y_o, \cdots Y_M\}$ with equal probability. Thus, the $K \cdot N$ components of the $\underline{X}$'s are zero mean i.i.D random variables. Our modified learning rule is

$$W_{ij} = \frac{1}{N} \sum_{l=1}^{K} X_i^l \cdot \left(\frac{1}{X_j^l}\right) \qquad (18)$$

Note that for $X_i \in \{+1, -1\}$ (18) is identical to (16).

Define

$$\widetilde{\Delta Y} \overset{\Delta}{=} \min_{i \neq j} |Y_i - Y_j|$$

$$A = \max_{i,j} \frac{|Y_i|^2}{|Y_j|}$$

We state that

## PROPOSITION:

The asymptotic capacity of the above network is given by

$$K = \frac{N}{\dfrac{16A^2}{(\widetilde{\Delta Y})^2} \log N} \tag{19}$$

## PROOF:

Define

$$P(K,N) = P_r \left\{ \begin{array}{l} K \text{ vectors chosen randomly as described} \\ \text{are stable states with the } W \text{ of ( )} \end{array} \right\}$$

$$P(K,N) = 1 - P_r(\bigcup A_{ij}) \geq 1 - \sum_{i,j} P_r(A_{ij}) \quad \begin{array}{l} i=1,\ldots,N \\ j=1,\ldots,K \end{array} \tag{20}$$

where $A_{ij}$ is the event that the $i$ th component of $j$ th vector is in error. We concentrate on the event $A_{11}$ W.L.G.

The input $u_1$ when $\underline{X}'$ is presented is given by

$$u_1 = \sum_{j=1}^{N} W_{1j} X_j^1 = X_1^1 + \frac{K-1}{N} X_1^1 + \frac{1}{N} \sum_{l=2}^{K} \sum_{j=2}^{N} X_1^l \frac{X_j^1}{X_j^l} \tag{21}$$

The first term is mapped by (17) into itself and corresponds to the desired signal.

The last term is a sum of $(K-1)(N-1)$ i.i.D zero mean random variables and corresponds to noise.

The middle term $\frac{K-1}{N}X_1^1$ is disposed of by assuming $\frac{K-1}{N}\underset{N\to\infty}{\to}0$. (With a zero diagonal

choice of $W$ (using (18) with $i\neq j$) this term does not appear).

$$P_r(\mathbf{A}_{11}) = P_r\{\text{ noise gets us out of range }\}$$

Denoting the noise by $I$ we have

$$P_r(\mathbf{A}_{11}) \le P_r(|I| > \frac{\tilde{\Delta Y}}{2}) \le 2\exp\left\{-\frac{\frac{1}{2}(\tilde{\Delta Y})^2 N^2}{(K-1)(N-1)4A^2}\right\} \tag{22}$$

where the first inequality is from the definition of $\tilde{\Delta Y}$ and the second uses the lemma of [6] p. 58. We thus

get

$$P(K,N) \ge 1 - K\cdot N\cdot 2\exp\left\{-\frac{(\tilde{\Delta Y})^2 N^2}{8(K-1)(N-1)A^2}\right\} \tag{23}$$

substituting (19) and taking $N\to\infty$ we get $P(K,N)\to1$ and this completes the proof.

## VII. DISCUSSION

Two classes of generalization of the Hopfield neural network model were presented. We give some remarks:

(a) Any combination of neurons from the two classes will have the convergence property as well.

(b) Our definition of the information capacity for the C.N.N. is useless since a full observation of the possible state transitions of the network is impossible.

## APPENDIX

We prove the following theorem.

Theorem

An upper bound on the number of multi threshold functions with $N$ inputs and $M$ points in the domain (out of $(n+1)^N$ possible points) $C_N^M$ is the solution of the recurrence relation

$$C_N^M = C_N^{M-1} + n \cdot C_{N-1}^{M-1} \tag{A.1}$$

Proof

Let us look on the $N$ dimensional weight space $\underline{W}$. Each input point $\underline{X}$ divides the weight space into $n+1$ regions by $n$ parallel hyperplanes $\sum\limits_{i=1}^{N} W_i X_i = t_k \quad k=1,...,n$. We keep adding points in such a way that the new $n$ hyperplanes corresponding to each added point partition the $\underline{W}$ space into as many regions as possible. Assume $M-1$ points have made $C_N^{M-1}$ regions and we add the $M$ 'th point. Each hyperplane (out of $n$) is divided into at most $C_{N-1}^{M-1}$ regions (being itself an $N-1$ dimensional space divided by $(M-1)n$ hyperlines). We thus have after passing the $n$ hyperplanes:

$$C_N^M = C_N^{M-1} + n \cdot C_{N-1}^{M-1}$$

is $C_N^M = (n+1) \sum\limits_{i=o}^{N-1} \binom{M-1}{i} n^i$ and the theorem is proved.

The solution of the recurrence in the case $M = (n+1)^N$ (all possible points) we have a bound on the number of multithreshold functions of $N$ variables equal to

$$C_n^{(n+1)^N} = (n+1) \sum\limits_{i=1}^{N-1} \binom{(n+1)^N - 1}{i} n^i \leq (n+1)^{N^2+N+1}$$

and the result used is established.

# LIST OF REFERENCES

[1] Hopfield J. J., "Neural networks and physical systems with emergent collective computational abilities", Proc. Nat. Acad. Sci. USA, Vol. 79 (1982), pp. 2554-2558.

[2] Abu-Mostafa Y.S. and Jacques J. St., "Information capacity of the Hopfield model", IEEE Trans. on Info. Theory, Vol. IT-31 (1985, pp. 461-464.

[3] Hopfield J. J., "Neurons with graded response have collective computational properties like those of two state neurons", Proc. Nat. Acad. Sci. USA, Vol. 81 (1984).

[4] Fleisher M., "Fast processing of autoregressive signals by a neural network", to be presented at IEEE Conference, Israel 1987.

[5] Levin, E., Private communication.

[6] Petrov, "Sums of independent random variables".
